# Learning Classification with Unlabeled Data

**Virginia R. de Sa**
desa@cs.rochester.edu
Department of Computer Science
University of Rochester
Rochester, NY 14627

## Abstract

One of the advantages of supervised learning is that the final error metric is available during training. For classifiers, the algorithm can directly reduce the number of misclassifications on the training set. Unfortunately, when modeling human learning or constructing classifiers for autonomous robots, supervisory labels are often not available or too expensive. In this paper we show that we can substitute for the labels by making use of structure between the pattern distributions to different sensory modalities. We show that minimizing the disagreement between the outputs of networks processing patterns from these different modalities is a sensible approximation to minimizing the number of misclassifications in each modality, and leads to similar results. Using the Peterson-Barney vowel dataset we show that the algorithm performs well in finding appropriate placement for the codebook vectors particularly when the confuseable classes are different for the two modalities.

## 1  INTRODUCTION

This paper addresses the question of how a human or autonomous robot can learn to classify new objects without experience with previous labeled examples. We represent objects with n-dimensional pattern vectors and consider piecewise-linear classifiers consisting of a collection of (labeled) codebook vectors in the space of the input patterns (See Figure 1). The classification boundaries are given by the voronoi tessellation of the codebook vectors. Patterns are said to belong to the class (given by the label) of the codebook vector to which they are closest.

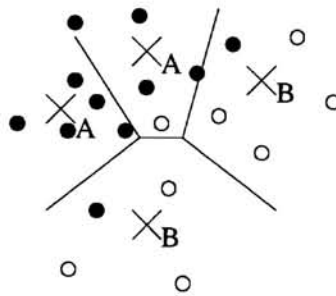

Figure 1: A piecewise-linear classifier in a 2-Dimensional input space. The circles represent data samples from two classes (filled (A) and not filled (B)). The X's represent codebook vectors (They are labeled according to their class A and B). Future patterns are classified according to the label of the closest codebook vector.

In [de Sa and Ballard, 1993] we showed that the supervised algorithm LVQ2.1[Kohonen, 1990] moves the codebook vectors to minimize the number of misclassified patterns. The power of this algorithm lies in the fact that it directly minimizes its final error measure (on the training set). The positions of the codebook vectors are placed not to approximate the probability distributions but to decrease the number of misclassifications.

Unfortunately in many situations labeled training patterns are either unavailable or expensive. The classifier can not measure its classification performance while learning (and hence not directly maximize it). One such unsupervised algorithm, Competitive Learning[Grossberg, 1976; Kohonen, 1982; Rumelhart and Zipser, 1986], has unlabeled codebook vectors that move to minimize a measure of the reconstruction cost. Even with subsequent labeling of the codebook vectors, they are not well suited for classification because they have not been positioned to induce optimal borders.

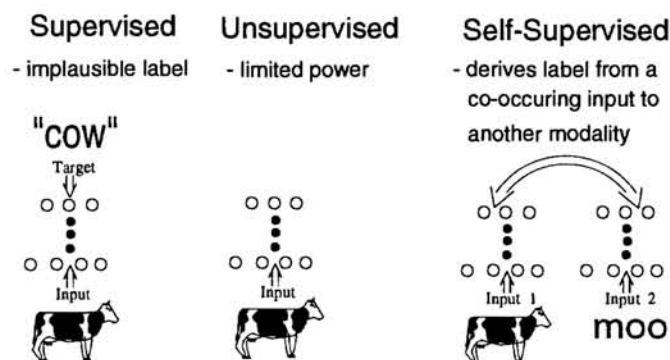

Figure 2: The idea behind the algorithm

This paper presents a new measure for piecewise-linear classifiers receiving unlabeled patterns from two or more sensory modalities. Minimizing the new measure is an approximation to minimizing the number of misclassifications directly. It takes advantage of the structure available in natural environments which results in sensations to different sensory modalities (and sub-modalities) that are correlated. For example, hearing "mooing" and

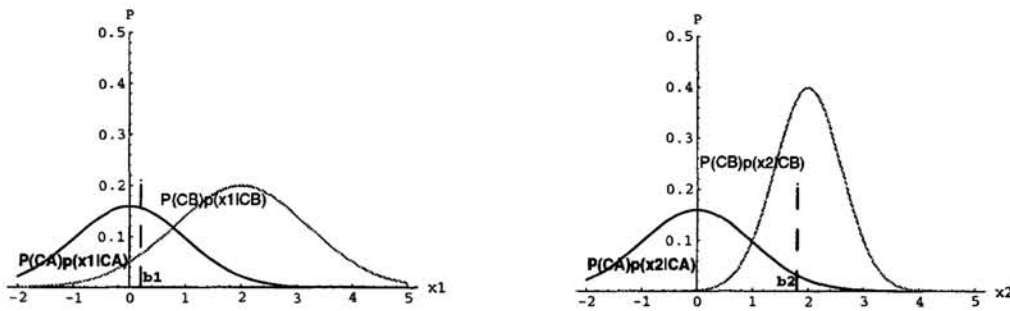

Figure 3: This figure shows an example world as sensed by two different modalities. If modality A receives a pattern from its Class A distribution, modality 2 receives a pattern from its own class A distribution (and the same for Class B). Without receiving information about which class the patterns came from, they must try to determine appropriate placement of the boundaries $b_1$ and $b_2$. $P(C_i)$ is the prior probability of Class $i$ and $p(x_j|C_i)$ is the conditional density of Class $i$ for modality $j$

seeing cows tend to occur together. So, although the sight of a cow does not come with an internal homuncular "cow" label it does co-occur with an instance of a "moo". The key is to process the "moo" sound to obtain a self-supervised label for the network processing the visual image of the cow and vice-versa. See Figure 2.

## 2   USING MULTI-MODALITY INFORMATION

One way to make use of the cross-modality structure is to derive labels for the codebook vectors (after they have been positioned either by random initialization or an unsupervised algorithm). The labels can be learnt with a competitive learning algorithm using a network such as that shown in Figure 4. In this network the hidden layer competitive neurons represent the codebook vectors. Their weights from the input neurons represent their positions in the respective input spaces. Presentation of the paired patterns results in activation of the closest codebook vectors in each modality (and 0's elsewhere). Co-occurring codebook vectors will then increase their weights to the same competitive output neuron. After several iterations the codebook vectors are given the (arbitrary) label of the output neuron to which they have the strongest weight. We will refer to this as the "labeling algorithm".

### 2.1   MINIMIZING DISAGREEMENT

A more powerful use of the extra information is for better placement of the codebook vectors themselves.

In [de Sa, 1994] we derive an algorithm that minimizes[1] the disagreement between the outputs of two modalities. The algorithm is originally derived not as a piecewise-linear classifier but as a method of moving boundaries for the case of two classes and an agent with two 1-Dimensional sensing modalities as shown in Figure 3.

Each class has a particular probability distribution for the sensation received by each modality. If modality 1 experiences a sensation from its pattern A distribution, modality 2 experiences a sensation from its own pattern A distribution. That is, the world presents patterns

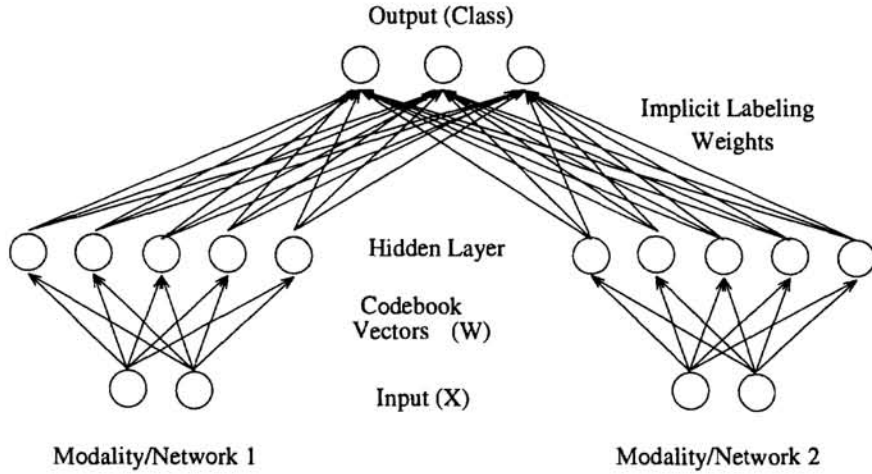

Figure 4: This figure shows a network for learning the labels of the codebook vectors. The weight vectors of the hidden layer neurons represent the codebook vectors while the weight vectors of the connections from the hidden layer neurons to the output neurons represent the output class that each codebook vector currently represents. In this example there are 3 output classes and two modalities each of which has 2-D input patterns and 5 codebook vectors.

from the 2-D joint distribution shown in Figure 5a) but each modality can only sample its 1-D marginal distribution (shown in Figure 3 and Figure 5a)).

We show [de Sa, 1994] that minimizing the disagreement error — the proportion of pairs of patterns for which the two modalities output different labels —

$$E(b_1, b_2) = Pr\{x_1 < b_1 \ \& \ x_2 > b_1\} + Pr\{x_1 > b_1 \ \& \ x_2 < b_2\} \tag{1}$$

$$E(b_1, b_2) = \int_{-\infty}^{b_1} \int_{b_2}^{\infty} f(x_1, x_2) dx_1 dx_2 + \int_{b_1}^{\infty} \int_{-\infty}^{b_2} f(x_1, x_2) dx_1 dx_2 \tag{2}$$

(where $f(x_1, x_2) = P(C_A)p(x_1|C_A)p(x_2|C_A) + P(C_B)p(x1|C_B)p(x_2|C_B)$ is the joint probability density for the two modalities) in the above problem results in an algorithm that corresponds to the optimal supervised algorithm except that the "label" for each modality's pattern is the hypothesized output of the other modality.

Consider the example illustrated in Figure 5. In the supervised case (Figure 5a)) the labels are given allowing sampling of the actual marginal distributions. For each modality, the number of misclassifications can be minimized by setting the boundaries for each modality at the crossing points of their marginal distributions.

However in the self-supervised system, the labels are not available. Instead we are given the output of the other modality. Consider the system from the point of view of modality 2. Its patterns are labeled according to the outputs of modality 1. This labels the patterns in Class A as shown in Figure 5b). Thus from the actual Class A patterns, the second modality sees the "labeled" distributions shown. Letting $a$ be the fraction of misclassified patterns from Class A, the resulting distributions are given by $(1 - a)P(C_A)p(x_2|C_A)$ and $(a)P(C_A)p(x_2|C_A)$.

Similarly Figure 5c) shows the effect on the patterns in class B. Letting $b$ be the fraction of Class B patterns misclassified, the distributions are given by $(1 - b)P(C_B)p(x_2|C_B)$

and $(b)P(C_B)p(x_2|C_B)$. Combining the effects on both classes results in the "labeled" distributions shown in Figure 5d). The "apparent Class A" distribution is given by $(1 - a)P(C_A)p(x_2|C_A) + (b)P(C_B)p(x_2|C_B)$ and the "apparent Class B" distribution by $(a)P(C_A)p(x_2|C_A) + (1 - b)P(C_B)p(x_2|C_B)$. Notice that even though the approximated distributions may be discrepant, if $a \approx b$, the crossing point will be close.

Simultaneously the second modality is labeling the patterns to the first modality. At each iteration of the algorithm both borders move according to the samples from the "apparent" marginal distributions.

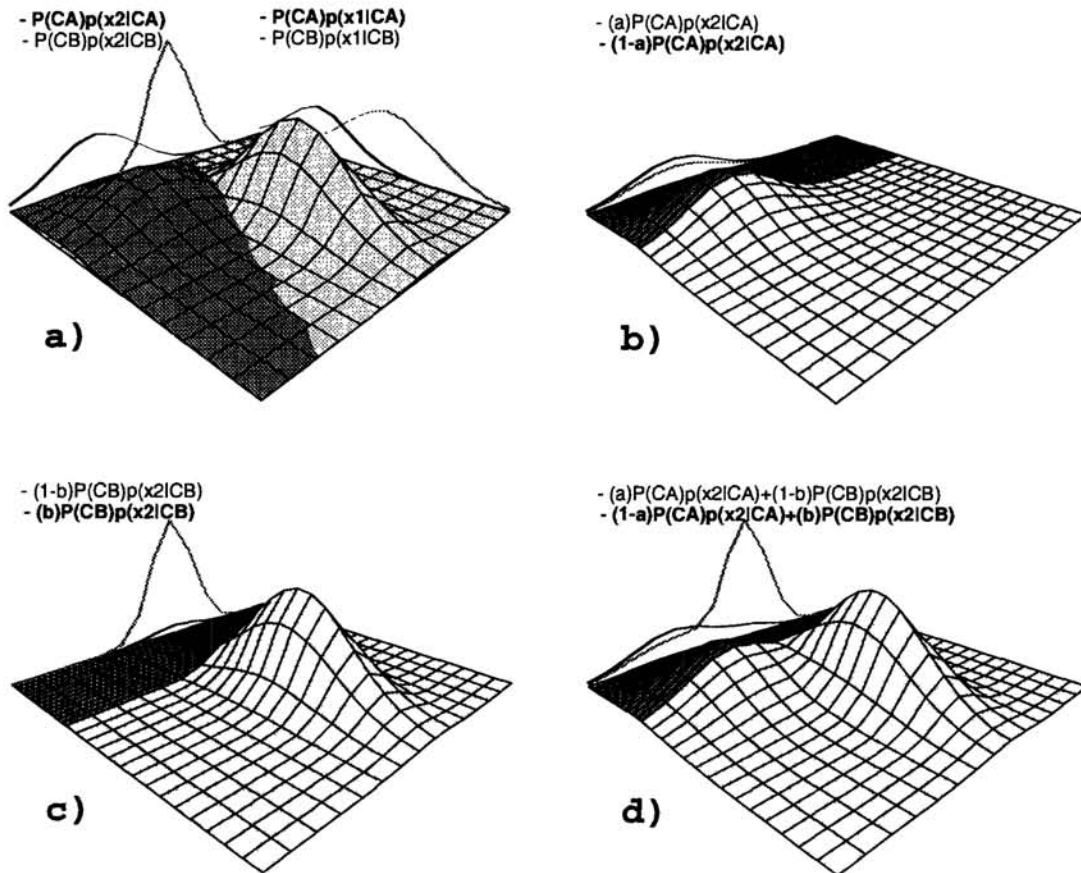

Figure 5: This figure shows an example of the joint and marginal distributions (For better visualization the scale of the joint distribution is twice that of the marginal distributions) for the example problem introduced in Figure 3. The darker gray represents patterns labeled "A", while the lighter gray are labeled "B". The dark and light curves are the corresponding marginal distributions with bold and regular labels respectively. a) shows the labeling for the supervised case. b),c) and d) reflect the labels given by modality 1 and the corresponding marginal distributions seen by modality 2. See text for more details

## 2.2   Self-Supervised Piecewise-Linear Classifier

The above ideas have been extended[de Sa, 1994] to rules for moving the codebook vectors in a piecewise-linear classifier. Codebook vectors are initially chosen randomly from the data patterns. In order to complete the algorithm idea, the codebook vectors need to be given initial labels (The derivation assumes that the current labels are correct). In LVQ2.1

the initial codebook vectors are chosen from among the data patterns that are consistent with their neighbours (according to a k-nearest neighbour algorithm); their labels are then taken as the labels of the data patterns. In order to keep our algorithm unsupervised the "labeling algorithm" mentioned earlier is used to derive labels for the initial codebook vectors.

Also due to the fact that the codebook vectors may cross borders or may not be accurately labeled in the initialization stage, they are updated throughout the algorithm by increasing the weight to the output class hypothesized by the other modality, from the neuron representing the closest codebook vector. The final algorithm is given in Figure 6

1. Randomly choose initial codebook vectors from data vectors

2. Initialize labels of codebook vectors using the labeling algorithm described in text

3. Repeat for each presentation of input patterns $X_1(n)$ and $X_2(n)$ to their respective modalities

   - find the two nearest codebook vectors in modality 1 -- $w_{1,i_1^*}, w_{1,i_2^*}$, and modality 2 -- $w_{2,k_1^*}, w_{2,k_2^*}$ to the respective input patterns
   - Find the hypothesized output class ($C_A$, $C_B$) in each modality (as given by the label of the closest codebook vector)
   - For each modality update the weights according to the following rules (Only the rules for modality 1 are given)
     If neither or both $w_{1,i_1^*}, w_{1,i_2^*}$ have the same label as $w_{2,k_1^*}$ or $X_1(n)$ does not lie within c(n) of the border between them no updates are done, otherwise

     $$w_{1,i^*}(n) = w_{1,i}^*(n-1) + \alpha(n) \frac{(X_1(n) - w_{1,i^*}(n-1))}{\|X_1(n) - w_{1,i^*}(n-1)\|}$$

     $$w_{1,j^*}(n) = w_{1,j}^*(n-1) - \alpha(n) \frac{(X_1(n) - w_{1,j^*}(n-1))}{\|X_1(n) - w_{1,j}^*(n-1)\|}$$

     where $w_{1,i^*}$ is the codebook vector with the same label, and $w_{1,j^*}$ is the codebook vector with another label.
   - update the labeling weights

Figure 6: The Self-Supervised piecewise-linear classifier algorithm

## 3 EXPERIMENTS

The following experiments were all performed using the Peterson and Barney vowel formant data [2]. The dataset consists of the first and second formants for ten vowels in a /hVd/ context from 75 speakers (32 males, 28 females, 15 children) who repeated each vowel twice [3].

To enable performance comparisons, each modality received patterns from the same dataset. This is because the final classification performance within a modality depends

Table 1: Tabulation of performance figures (mean percent correct and sample standard deviation over 60 trials and 2 modalities). The heading $i - j$ refers to performance measured after the $j^{th}$ step during the $i^{th}$ iteration. (Note Step 1 is not repeated during the multi-iteration runs).

|  | 1-2 | 1-3 | 2-2 | 2-3 | 3-3 | 4-3 | 5-3 |
|---|---|---|---|---|---|---|---|
| same-paired vowels | $60 \pm 5$ | $75 \pm 4$ | $73 \pm 4$ | $76 \pm 4$ | $76 \pm 4$ | $76 \pm 4$ | $76 \pm 4$ |
| random pairing | $60 \pm 4$ | $77 \pm 3$ | $77 \pm 3$ | $79 \pm 2$ | $79 \pm 2$ | $79 \pm 2$ | $79 \pm 2$ |

not only on the difficulty of the measured modality but also on that of the other "labeling" modality. Accuracy was measured individually (on the training set) for both modalities and averaged. These results were then averaged over 60 runs. The results described below are also tabulated in Table 1

In the first experiment, the classes were paired so that the modalities received patterns from the same vowel class. If modality 1 received an [a] vowel, so did modality 2 and likewise for all the vowel classes (i.e. $p(x_1|C_j) = p(x_2|C_j)$ for all $j$). After the labeling algorithm stage, the accuracy was $60 \pm 5\%$ as the initial random placement of the codebook vectors does not induce a good classifier. After application of the third step in Figure 6 (the minimizing-disagreement part of the algorithm) the accuracy was $75 \pm 4\%$. At this point the codebook vectors are much better suited to defining appropriate classification boundaries.

It was discovered that all stages of the algorithm tended to produce better results on the runs that started with better random initial configurations. Thus, for each run, steps 2 and 3 were repeated with the final codebook vectors. Average performance improved ( $73 \pm 4\%$ after step 2 and $76 \pm 4\%$ after step 3). Steps 2 and 3 were repeated several more times with no further significant increase in performance.

The power of using the cross-modality information to move the codebook vectors can be seen by comparing these results to those obtained with unsupervised competitive learning within modalities followed by an optimal supervised labeling algorithm which gave a performance of 72%.

One of the features of multi-modality information is that classes that are easily confuseable in one modality may be well separated in another. This should improve the performance of the algorithm as the "labeling" signal for separating the overlapping classes will be more reliable. In order to demonstrate this, more tests were conducted with random pairing of the vowels for each run. For example presentation of [a] vowels to one modality would be paired with presentation of [i] vowels to the other. That is $p(x_1|C_j) = p(x_2|C_{\alpha_j})$ for a random permutation $\alpha_1, \alpha_2..\alpha_{10}$. For the labeling stage the performance was as before ($60 \pm 4\%$) as the difficulty within each modality has not changed. However after the minimizing-disagreement algorithm the results were better as expected. After 1 and 2 iterations of the algorithm, $77 \pm 3\%$ and $79 \pm 2\%$ were classified correctly. These results are close to those obtained with the related supervised algorithm LVQ2.1 of 80%.

## 4   DISCUSSION

In summary, appropriate classification borders can be learnt without an explicit external labeling or supervisory signal. For the particular vowel recognition problem, the perfor-mance of this "self-supervised" algorithm is almost as good as that achieved with super-

vised algorithms. This algorithm would be ideal for tasks in which signals for two or more modalities are available, but labels are either not available or expensive to obtain.

One specific task is learning to classify speech sounds from images of the lips and the acoustic signal. Stork et. al. [1992] performed this task with a supervised algorithm but one of the main limitations for data collection was the manual labeling of the patterns [David Stork, personal communication, 1993]. This task also has the feature that the speech sounds that are confuseable are not confuseable visually and vice-versa [Stork *et al.*, 1992]. This complementarity helps the performance of this classifier as the other modality provides more reliable labeling where it is needed most.

The algorithm could also be used for learning to classify signals to a single modality where the signal to the other "modality" is a temporally close sample. As the world changes slowly over time, signals close in time are likely from the same class. This approach should be more powerful than that of [Földiák, 1991] as signals close in time need not be mapped to the same codebook vector but the closest codebook vector of the same class.

## Acknowledgements

I would like to thank Steve Nowlan for making the vowel formant data available to me. Many thanks also to Dana Ballard, Geoff Hinton and Jeff Schneider for their helpful conversations and suggestions. A preliminary version of parts of this work appears in greater depth in [de Sa, 1994].

## Footnotes

[1]the goal is actually to find a non-trivial local minimum (for details see [de Sa, 1994])

[2]obtained from Steven Nowlan

[3]3 speakers were missing one vowel and the raw data was linearly transformed to have zero mean and fall within the range [−3, 3] in both components

## References

[de Sa, 1994]  Virginia R. de Sa, "Minimizing disagreement for self-supervised classification," In M.C. Mozer, P. Smolensky, D.S. Touretzky, J.L. Elman, and A.S. Weigend, editors, *Proceedings of the 1993 Connectionist Models Summer School*, pages 300—307. Erlbaum Associates, 1994.

[de Sa and Ballard, 1993]  Virginia R. de Sa and Dana H. Ballard, "a note on learning vector quantization," In C.L. Giles, S.J.Hanson, and J.D. Cowan, editors, *Advances in Neural Information Processing Systems 5*, pages 220—227. Morgan Kaufmann, 1993.

[Földiák, 1991]  Peter Földiák, "Learning Invariance from Transformation Sequences," *Neural Computation*, 3(2):194–200, 1991.

[Grossberg, 1976]  Stephen Grossberg, "Adaptive Pattern Classification and Universal Recoding: I. Parallel Development and Coding of Neural Feature Detectors," *Biological Cybernetics*, 23:121–134, 1976.

[Kohonen, 1982]  Teuvo Kohonen, "Self-Organized Formation of Topologically Correct Feature Maps," *Biological Cybernetics*, 43:59–69, 1982.

[Kohonen, 1990]  Teuvo Kohonen, "Improved Versions of Learning Vector Quantization," In *IJCNN International Joint Conference on Neural Networks*, volume 1, pages I–545–I–550, 1990.

[Rumelhart and Zipser, 1986]  D. E. Rumelhart and D. Zipser, "Feature Discovery by Competitive Learning," In David E. Rumelhart, James L. McClelland, and the PDP Research Group, editors, *Parallel Distributed Processing: Explorations in the Microstructure of Cognition*, volume 2, pages 151–193. MIT Press, 1986.

[Stork *et al.*, 1992]  David G. Stork, Greg Wolff, and Earl Levine, "Neural network lipreading system for improved speech recognition," In *IJCNN International Joint Conference on Neural Networks*, volume 2, pages II–286—II–295, 1992.